# A Model of Spatial Representations in Parietal Cortex Explains Hemineglect

**Alexandre Pouget**
Dept of Neurobiology
UCLA
Los Angeles, CA 90095-1763
alex@salk.edu

**Terrence J. Sejnowski**
Howard Hughes Medical Institute
The Salk Institute
La Jolla, CA 92037
terry@salk.edu

## Abstract

We have recently developed a theory of spatial representations in which the position of an object is not encoded in a particular frame of reference but, instead, involves neurons computing basis functions of their sensory inputs. This type of representation is able to perform nonlinear sensorimotor transformations and is consistent with the response properties of parietal neurons. We now ask whether the same theory could account for the behavior of human patients with parietal lesions. These lesions induce a deficit known as hemineglect that is characterized by a lack of reaction to stimuli located in the hemispace contralateral to the lesion. A simulated lesion in a basis function representation was found to replicate three of the most important aspects of hemineglect: i) The models failed to cross the leftmost lines in line cancellation experiments, ii) the deficit affected multiple frames of reference and, iii) it could be object centered. These results strongly support the basis function hypothesis for spatial representations and provide a computational theory of hemineglect at the single cell level.

## 1  Introduction

According to current theories of spatial representations, the positions of objects are represented in multiple modules throughout the brain, each module being specialized for a particular sensorimotor transformation and using its own frame of reference. For instance, the lateral intraparietal area (LIP) appears to encode the location of objects in oculocentric coordinates, presumably for the control of saccadic eye movements. The ventral intraparietal cortex (VIP) and the premotor cortex, on the other hand, seem to use head-centered coordinates and might be

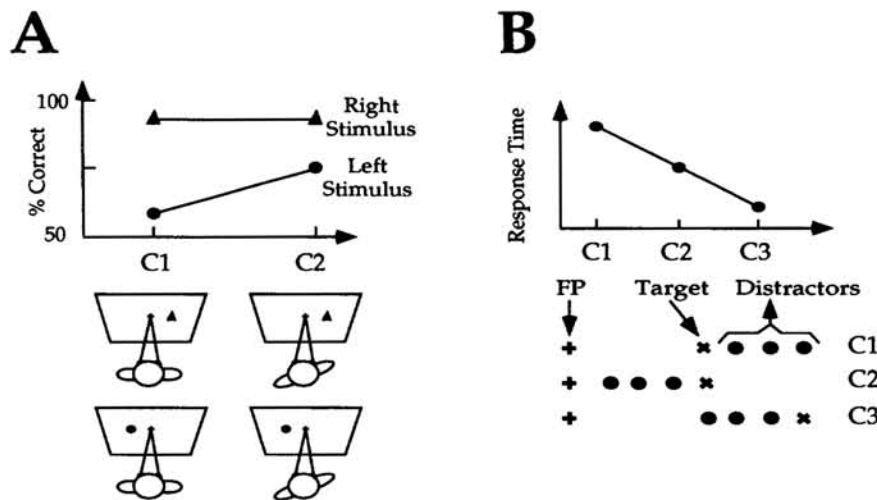

Figure 1: A. Retinotopic neglect modulated by egocentric position. B. Stimulus-centered neglect

involved in the control of hand movements toward the face.

This modular theory of spatial representations is not fully consistent with the behavior of patients with parietal or frontal lesions. Such lesions causes a syndrome known as hemineglect which is characterized by a lack of response to sensory stimuli appearing in the hemispace contralateral to the lesion [3]. According to the modular view, the deficit should be behavior dependent, e.g., oculocentric for eye movements, head-centered for reaching. However, experimental and clinical studies show that this is not the case. Instead, neglect affects multiple frames of reference simultaneously, and to a first approximation, independently of the task.

This point is particularly clear in an experiment by Karnath et al (1993) (Figure 1A). Subjects were asked to identify a stimulus that can appear on either side of the fixation point. In order to test whether the position of the stimuli with respect to the body affects performance, two conditions were tested: a control condition with head straight ahead (C1), and a second condition with head rotated 20 degrees on the right (or equivalently, with the trunk rotated 20 degrees on the left, see figure) (C2). In C2, both stimuli appeared further to the right of the trunk while being at the same location with respect to the head and retina than in C1. Moreover, the trunk-centered position of the left stimulus in C2 was the same than the trunk-centered position of the right stimulus in C1.

As expected, subjects with right parietal lesions performed better on the right stimulus in the control condition, a result consistent with both, retinotopic and trunk-centered neglect. To distinguish between the two frames of reference, one needs to compare performance across conditions.

If the deficit is purely retinocentric, the results should be identical in both conditions, since the retinotopic location of the stimuli does not vary. If, on the other hand, the deficit is purely trunk-centered, the performance on the left stimulus should improve when the head is turned right since the stimulus now appears further toward the right of the trunk-centered hemispace. Furthermore, performance on the right stimulus in the control condition should be the same as performance on the left stimulus in the rotated condition, since they share the same trunk-centered position in both cases.

Neither of these hypotheses can fully account for the data. As expected from a retinotopic neglect, subjects always performed better on the right stimulus in both conditions. However, performance on the left stimulus improved when the head was turned right (C2), though not sufficiently to match the level of performance on the right stimulus in the control condition (C1). Therefore, these results suggest a retinotopic neglect modulated by trunk-centered factors.

In addition, Karnath et al (1991) tested patients on a similar experiment in which subjects were asked to generate a saccade toward the target. The analysis of reaction time revealed the same type of results than the one found in the identification task, thereby demonstrating that the spatial deficit is, to a first approximation, independent of the task.

An experiment by Arguin and Bub (1993) suggests that neglect can be object-centered as well. As shown in figure 1B, they found that reaction times were faster when the target appeared on the right of a set of distractors (C2), as opposed to the left (C1), even though the target is at the same retinotopic location in both conditions. Interestingly, moving the target further to the right leads to even faster reaction times (C3), showing that hemineglect is not only object-centered but retinotopic as well in this task.

These results strongly support the existence of spatial representations using multiple frames of reference simultaneously shared by several behaviors. We have recently developed a theory [6] which has precisely these properties and we ask here whether a simulated lesion would lead to a deficit similar to hemineglect. Our theory posits that parietal neurons computes basis function (BF) of sensory signals, such as visual, or auditory inputs, and posture signals, such as eye or head position. The resulting representation, which we called a basis function map, can be used for performing nonlinear transformations of the sensory inputs, the type of transformations required for sensorimotor coordination.

## 2 Model Organization

The model contains two distinct parts: a network for performing sensorimotor transformations and a selection mechanism.

### 2.1 Network Architecture

We implemented a network using basis function units in the intermediate layer to perform a transformation from a visual retinotopic map to two motor maps in, respectively, head-centered and oculocentric coordinates (Figure 2). The input contains a retinotopic visual map analog to the one found in the early stages of visual processing, and a set of units encoding eye position, similar to the neurons found in the intralaminar nucleus of the thalamus. These input units project to a set of intermediate units shared by both transformations. Each intermediate unit computes a gaussian of the retinal location of object, $r_x$, multiplied by a sigmoid of eye position, $e_x$:

$$o_i = \frac{e^{-\frac{(r_x - r_{xi})^2}{2\sigma^2}}}{1 + e^{-\frac{e_x - e_{xi}}{t}}} \tag{1}$$

These units are organized in a map covering all possible combinations of retinal and eye position selectivities. As we have shown elsewhere [6], this type of response function is consistent with the response of single parietal neurons found in area 7a.

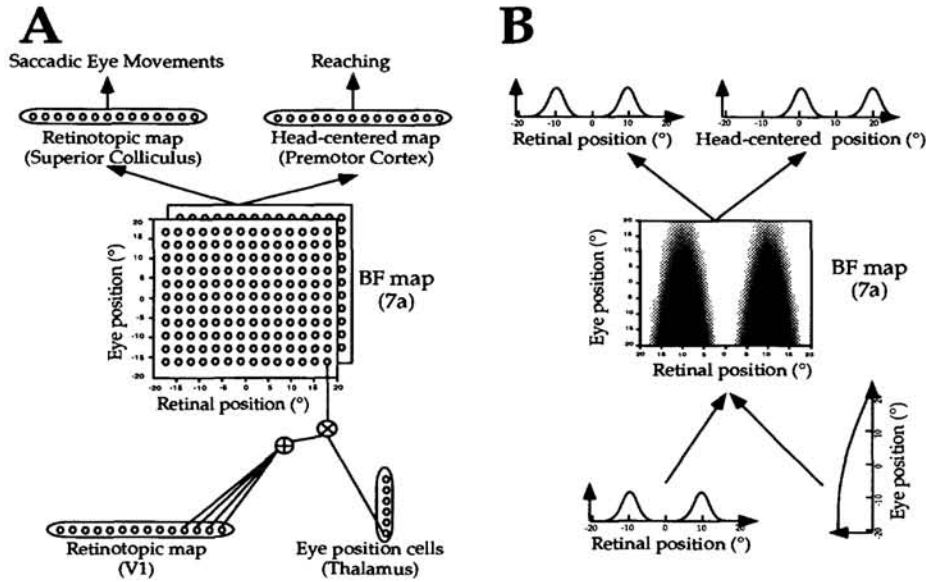

Figure 2: A. Network architecture B. Typical pattern of activity

The resulting map forms a basis function map which encodes the location of objects in head-centered and retinotopic coordinates simultaneously.

The activity of the unit in the output maps is computed by a simple linear combination of the BF unit activities. Appropriate values of the weights were found by using linear regression techniques.

This architecture mimics the pattern of projections of the parietal area 7a. 7a is known to project to, both, the superior colliculus and the premotor cortex (via the ventral parietal area, VIP), in which neurons have, respectively, retinotopic and head-centered visual receptive fields. Figure 2B shows a typical pattern of activity in the network when two stimuli are presented simultaneously while the eye fixated 10 degrees toward the right.

## 2.2   Hemispheric Biases and Lesion Model

Neurophysiological data indicate that both hemispheres contain neurons with all possible combinations of retinal and eye position selectivities, but with a contralateral bias. Hence, most neurons in the right parietal cortex (resp. left) have their retinal receptive field on the left hemiretina (resp. right). The bias for eye position is much weaker but a trend has been reported in several studies [1].

Therefore, spatial representations in a patient with a right parietal lesions are biased toward the right side of space. We modeled such a lesion by using a similar bias in the intermediate layer of our network. The BF map simply has more neurons tuned to *right retinal* and *eye positions*. We found that the exact profile of the neuronal gradient across the basis function maps did not matter as long as it was monotonic and contralateral for both eye position and retinal location.

## 2.3   Selection model

We also developed a selection mechanism to model the behavior of patients when presented with several stimuli simultaneously. The simultaneous presentation of

stimuli induces multiple hills of activity in the network (see for instance the pattern of activity shown in figure 1B for two visual stimuli). Our selection mechanism operates on the peak values of these hills.

At each time step, the most active stimulus is selected according to a winner-take-all and its corresponding activity is set to zero (inhibition of return). At the next time step, the second highest stimuli is selected while the previously selected item is allowed to recover slowly. This procedure ensures that the most active item is not selected twice in a row, but because of the recovery process, stimulus with high activity might be selected again if displayed long enough.

This mechanism is such that the probability of selecting an item is proportional to two factors: the absolute amount of activity associated with the item, and the relative activity with respect to other competing items.

## 2.4   Evaluating network performance

We used this model to simulate several experiments in which patient performance was evaluated according to reaction time or percent of correct response.

Reaction time in the model was taken to be proportional to the number of time steps required by our selection mechanism to select a particular target. Performance on identification task was assumed to be proportional to the strength of the activity generated by the stimuli in the BF map.

# 3   Results

## 3.1   Line cancellation

We first tested the network on the line cancellation test, a test in which patients are asked to cross out short line segments uniformly spread over a page. To simulate this test, we presented the display shown in figure 3A and we ran the selection mechanism to determine which lines get selected by the network. As illustrated in figure 3A, the network crosses out only the lines located in the right half of the display, just as left neglect patients do in the same task. The rightward gradient introduced by the lesion biases the selection mechanism in favor of the most active lines, i.e., the ones on the right. As a result, the rightmost lines win the competition over and over, preventing the network from selecting the left lines.

## 3.2   Mixture of frames of reference

Next, we sought to determine the frame of reference of neglect in the model. Since Karnath et al (1993) manipulated head position, we simulated their experiment by using a BF map integrating visual inputs with head position, rather than eye position. We show in figure 3B the pattern of activity obtained in the retinotopic output layer of the network in the various experimental conditions (the other maps behaved in a similar way). In both conditions, head straight ahead (dotted lines) or turned on the side (solid lines), the right stimulus is associated with more activity than the left stimulus. This is the consequence of the larger number of cells in the basis function map for rightward position. In addition, the activity for the left stimulus increases when the head is turned to the right. This effect is related to the larger number of cells in the basis function maps tuned to right head positions.

Since network performance is proportional to activity strength, the overall pattern of performance was found to be similar to what has been reported in human patients

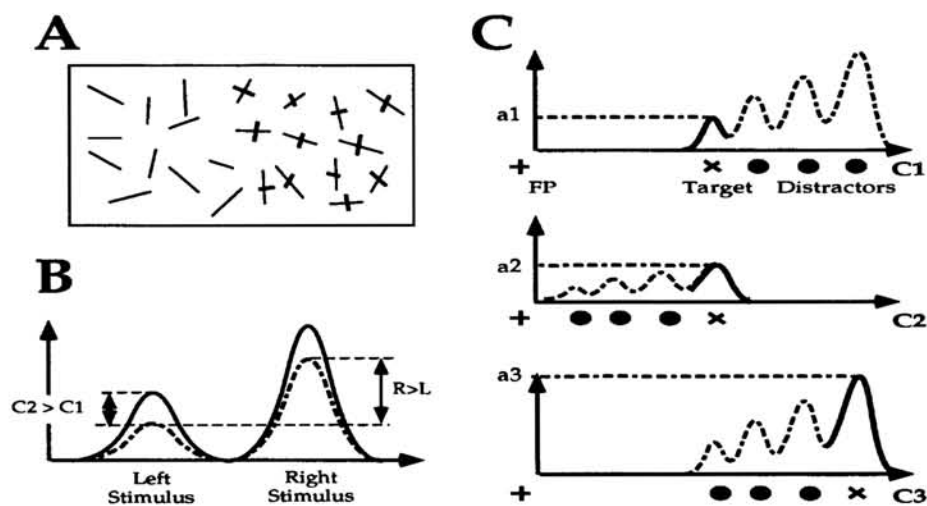

Figure 3: Network behavior in line cancellation task (A). Activity patterns in the retinotopic output layer when simulating the experiments by Karnath et al (1993) (B) and Arguin et al (1993) (C)

(figure 1A), namely: the right stimulus was better processed than the left stimulus and performance on the left stimulus increases when the head is rotated toward the right. Therefore, just like in human, neglect in the model is neither retinocentric nor trunk-centered alone, but both at the same time.

## 3.3  Object-centered effect

When simulating Arguin et al (1993) experiments, the network reaction times were found to follow the same trends than for human patients. Figure 3C illustrates the patterns of activity in the retinotopic output layer of the network when simulating the three conditions of Arguin experiments. Notice that the absolute activity associated with the target (solid lines) in conditions 1 and 2 is the same, but the activity of the distractors (dotted lines) differs in the two conditions. In condition 1, they have higher relative activity and thereby strongly delay the detection of the target by the selection mechanism. In condition 2, the distractors are now less active than the target and do not delay target processing as much as they do in condition 1. The reaction time decreases even more in condition 3, due to a higher absolute activity associated with the target. Therefore, the network exhibits retinocentric and object-centered neglect, just like parietal patients [2].

## 4  Discussion

The model of parietal cortex presented here was originally developed by considering the response properties of parietal neurons and the computational constraints inherent in sensorimotor transformations. It was not designed to model neglect, so its ability to account for a wide range of deficits is additional evidence in favor of the basis function hypothesis.

As we have shown, our model captures three essential aspects of the neglect syndrome: 1) It reproduces the pattern of line crossing reported in patients in line-cancellation experiments, 2) the deficit coexists in multiple frames of reference simultaneously, and 3) the model accounts for some of the object-based effects.

We can account for a very large number of studies beyond the ones we have considered here, using very similar computational principles. We can reproduce, in particular, the behavior of patients in line-bisection experiments and we can explain why neglect affects multiple cartesian frames of reference such as retinotopic, head-centered, trunk-centered, environment-centered (i.e. with respect to gravity), and object-centered.

It must be emphasized that these results have been obtained without using explicit representations of these various cartesian frames of reference (except for the retinotopy of the BF map). In fact, this is precisely because the lesion affected noncartesian representations that we have been able to reproduce these results. We have assumed that the lesion affects the functional space in which the basis functions are defined. This functional space shares common dimensions with cartesian spaces, but cannot be reduced to the latter. Hence, a basis function map integrating retinal location and head position is retinotopic, but not solely retinotopic. Consequently, any attempts to determine the cartesian space in which hemineglect operates is bound to lead to inconclusive results in which cartesian frames of reference appear to be mixed.

This study and previous research [6] suggests that the parietal cortex represents the position of objects by computing basis functions of the sensory and posture inputs. It would now be interesting to see if this hypothesis could also account for sensorimotor adaptation, such as learning to reach properly when wearing visual prisms. We predict that adaptation takes place in several frames of reference simultaneously, a prediction that is testable and would provide further support for the basis function framework.

# References

[1] R.A. Andersen, C. Asanuma, G. Essick, and R.M. Siegel. Corticocortical connections of anatomically and physiologically defined subdivisions within the inferior parietal lobule. *Journal of Comparative Neurology*, 296(1):65–113, 1990.

[2] M. Arguin and D.N. Bub. Evidence for an independent stimulus-centered reference frame from a case of visual hemineglect. *Cortex*, 29:349–357, 1993.

[3] K.M. Heilman, R.T. Watson, and E. Valenstein. Neglect and related disorders. In K.M. Heilman and E. Valenstein, editors, *Clinical Neuropsychology*, pages 243–294. Oxford University Press, New York, 1985.

[4] H.O. Karnath, K. Christ, and W. Hartje. Decrease of contralateral neglect by neck muscle vibration and spatial orientation of trunk midline. *Brain*, 116:383–396, 1993.

[5] H.O. Karnath, P. Schenkel, and B. Fischer. Trunk orientation as the determining factor of the 'contralateral' deficit in the neglect syndrome and as the physical anchor of the internal representation of body orientation in space. *Brain*, 114:1997–2014, 1991.

[6] A. Pouget and T.J. Sejnowski. Spatial representations in the parietal cortex may use basis functions. In G. Tesauro, D.S. Touretzky, and T.K. Leen, editors, *Advances in Neural Information Processing Systems*, volume 7. MIT Press, Cambridge, MA, 1995.